# Approximate Inference by Compilation to Arithmetic Circuits

**Daniel Lowd**
Department of Computer and Information Science
University of Oregon
Eugene, OR 97403-1202
lowd@cs.uoregon.edu

**Pedro Domingos**
Department of Computer Science and Engineering
University of Washington
Seattle, WA 98195-2350
pedrod@cs.washington.edu

## Abstract

Arithmetic circuits (ACs) exploit context-specific independence and determinism to allow exact inference even in networks with high treewidth. In this paper, we introduce the first ever *approximate* inference methods using ACs, for domains where exact inference remains intractable. We propose and evaluate a variety of techniques based on exact compilation, forward sampling, AC structure learning, Markov network parameter learning, variational inference, and Gibbs sampling. In experiments on eight challenging real-world domains, we find that the methods based on sampling and learning work best: one such method ($AC^2$-F) is faster and usually more accurate than loopy belief propagation, mean field, and Gibbs sampling; another ($AC^2$-G) has a running time similar to Gibbs sampling but is consistently more accurate than all baselines.

## 1  Introduction

Compilation to arithmetic circuits (ACs) [1] is one of the most effective methods for exact inference in Bayesian networks. An AC represents a probability distribution as a directed acyclic graph of addition and multiplication nodes, with real-valued parameters and indicator variables at the leaves. This representation allows for linear-time exact inference in the size of the circuit. Compared to a junction tree, an AC can be exponentially smaller by omitting unnecessary computations, or by performing repeated subcomputations only once and referencing them multiple times. Given an AC, we can efficiently condition on evidence or marginalize variables to yield a simpler AC for the conditional or marginal distribution, respectively. We can also compute all marginals in parallel by differentiating the circuit. These many attractive properties make ACs an interesting and important representation, especially when answering many queries on the same domain. However, as with junction trees, compiling a BN to an equivalent AC yields an exponentially-sized AC in the worst case, preventing their application to many domains of interest.

In this paper, we introduce *approximate* compilation methods, allowing us to construct effective ACs for previously intractable domains. For selecting circuit structure, we compare exact compilation of a simplified network to learning it from samples. Structure selection is done once per domain, so the cost is amortized over all future queries. For selecting circuit parameters, we compare variational inference to maximum likelihood learning from samples. We find that learning from samples works

best for both structure and parameters, achieving the highest accuracy on eight challenging, real-world domains. Compared to loopy belief propagation, mean field, and Gibbs sampling, our $\text{AC}^2$-F method, which selects parameters once per domain, is faster and usually more accurate. Our $\text{AC}^2$-G method, which optimizes parameters at query time, achieves higher accuracy on every domain with a running time similar to Gibbs sampling.

The remainder of this paper is organized as follows. In Section 2, we provide background on Bayesian networks and arithmetic circuits. In Section 3, we present our methods and discuss related work. We evaluate the methods empirically in Section 4 and conclude in Section 5.

## 2 Background

### 2.1 Bayesian networks

*Bayesian networks* (BNs) exploit conditional independence to compactly represent a probability distribution over a set of variables, $\{X_1, \ldots, X_n\}$. A BN consists of a directed, acyclic graph with a node for each variable, and a set of conditional probability distributions (CPDs) describing the probability of each variable, $X_i$, given its parents in the graph, denoted $\pi_i$ [2]. The full probability distribution is the product of the CPDs: $P(X) = \prod_{i=1}^{n} P(X_i|\pi_i)$.

Each variable in a BN is conditionally independent of its non-descendants given its parents. Depending on the how the CPDs are parametrized, there may be additional independencies. For discrete domains, the simplest form of CPD is a conditional probability table, but this requires space exponential in the number of parents of the variable. A more scalable approach is to use decision trees as CPDs, taking advantage of context-specific independencies [3, 4, 5]. In a decision tree CPD for variable $X_i$, each interior node is labeled with one of the parent variables, and each of its outgoing edges is labeled with a value of that variable. Each leaf node is a multinomial representing the marginal distribution of $X_i$ conditioned on the parent values specified by its ancestor nodes and edges in the tree.

Bayesian networks can be represented as log-linear models:
$$\log P(X = x) = -\log Z + \sum_i w_i f_i(x) \tag{1}$$
where each $f_i$ is a *feature*, each $w_i$ is a real-valued *weight*, and $Z$ is the partition function. In BNs, $Z$ is 1, since the conditional distributions ensure global normalization. After conditioning on evidence, the resulting distribution may no longer be a BN, but it can still be represented as a log linear model.

The goal of inference in Bayesian networks and other graphical models is to answer arbitrary marginal and conditional queries (*i.e.*, to compute the marginal distribution of a set of query variables, possibly conditioned on the values of a set of evidence variables). Popular methods include variational inference, Gibbs sampling, and loopy belief propagation.

In variational inference, the goal is to select a tractable distribution $Q$ that is as close as possible to the original, intractable distribution $P$. Minimizing the KL divergence from $P$ to $Q$ ($\text{KL}(P \| Q)$) is generally intractable, so the "reverse" KL divergence is typically used instead:
$$\text{KL}(Q \| P) = \sum_x Q(x) \log \frac{Q(x)}{P(x)} = -H_Q(x) - \sum_i w_i E_Q[f_i] + \log Z_P \tag{2}$$
where $H_Q(x)$ is the entropy of $Q$, $E_Q$ is an expectation computed over the probability distribution $Q$, $Z_P$ is the partition function of $P$, and $w_i$ and $f_i$ are the weights and features of $P$ (see Equation 1). This quantity can be minimized by fixed-point iteration or by using a gradient-based numerical optimization method. What makes the reverse KL divergence more tractable to optimize is that the expectations are done over $Q$ instead of $P$. This minimization also yields bounds on the log partition function, or the probability of evidence in a BN. Specifically, because $\text{KL}(Q \| P)$ is non-negative, $\log Z_P \geq H_Q(x) + \sum_i w_i E_Q[f_i]$.

The most commonly applied variational method is mean field, in which $Q$ is chosen from the set of fully factorized distributions. Generalized or structured mean field operates on a set of clusters (possibly overlapping), or junction tree formed from a subset of the edges [6, 7, 8]. Selecting the best tractable substructure is a difficult problem. One approach is to greedily delete arcs until the junction tree is tractable [6]. Alternately, Xing *et al.* [7] use weighted graph cuts to select clusters for structured mean field.

## 2.2 Arithmetic circuits

The probability distribution represented by a Bayesian network can be equivalently represented by a multilinear function known as the *network polynomial* [1]: $P(X_1 = x_1, \ldots, X_n = x_n) = \sum_{\mathbf{X}} \prod_{i=1}^{n} I(X_i = x_i) P(X_i = x_i | \Pi_i = \pi_i)$ where the sum ranges over all possible instantiations of the variables, $I()$ is the indicator function (1 if the argument is true, 0 otherwise), and the $P(X_i | \Pi_i)$ are the parameters of the BN. The probability of any partial instantiation of the variables can now be computed simply by setting to 1 all the indicators consistent with the instantiation, and to 0 all others. This allows arbitrary marginal and conditional queries to be answered in time linear in the size of the polynomial. Furthermore, differentiating the network with respect to its weight parameters $(w_i)$ yields the probabilities of the corresponding features $(f_i)$.

The size of the network polynomial is exponential in the number of variables, but it can be more compactly represented using an *arithmetic circuit* (AC). An AC is a rooted, directed acyclic graph whose leaves are numeric constants or variables, and whose interior nodes are addition and multiplication operations. The value of the function for an input tuple is computed by setting the variable leaves to the corresponding values and computing the value of each node from the values of its children, starting at the leaves. In the case of the network polynomial, the leaves are the indicators and network parameters. The AC avoids the redundancy present in the network polynomial, and can be exponentially more compact.

Every junction tree has a corresponding AC, with an addition node for every instantiation of a separator, a multiplication node for every instantiation of a clique, and a summation node as the root. Thus one way to compile a BN into an AC is via a junction tree. However, when the network contains context-specific independences, a much more compact circuit can be obtained. Darwiche [1] describes one way to do this, by encoding the network into a special logical form, factoring the logical form, and extracting the corresponding AC.

Other exact inference methods include variable elimination with algebraic decision diagrams (which can also be done with ACs [9]), AND/OR graphs [10], bucket elimination [11], and more.

## 3 Approximate Compilation of Arithmetic Circuits

In this section, we describe AC$^2$ (Approximate Compilation of Arithmetic Circuits), an approach for constructing an AC to approximate a given BN. AC$^2$ does this in two stages: structure search and parameter optimization. The structure search is done in advance, once per network, while the parameters may be selected at query time, conditioned on evidence. This amortizes the cost of the structure search over all future queries.The parameter optimization allows us to fine-tune the circuit to specific pieces of evidence. Just as in variational inference methods such as mean field, we optimize the parameters of a tractable distribution to best approximate an intractable one. Note that, if the BN could be compiled exactly, this step would be unnecessary, since the conditional distribution would always be optimal.

### 3.1 Structure search

We considered two methods for generating circuit structures. The first is to prune the BN structure and then compile the simplified BN exactly. The second is to approximate the BN distribution with a set of samples and learn a circuit from this pseudo-empirical data.

### 3.1.1 Pruning and compiling

Pruning and compiling a BN is somewhat analogous to edge deletion methods (e.g., [6]), except that instead of removing entire edges and building the full junction tree, we introduce context-specific independencies and build an arithmetic circuit that can exploit them. This finer-grained simplification offers the potential of much richer models or smaller circuits. However, it also offers more challenging search problems that must be approximated heuristically.

We explored several techniques for greedily simplifying a network into a tractable AC by pruning splits from its decision-tree CPDs. Ideally, we would like to have bounds on the error of our simplified model, relative to the original. This can be accomplished by bounding the ratio of each log con-

ditional probability distribution, so that the approximated log probability of every instance is within a constant factor of the truth, as done by the Multiplicative Approximation Scheme (MAS) [12]. However, we found that the bounds for our networks were very large, with ratios in the hundreds or thousands. This occurs because our networks have probabilities close to 0 and 1 (with logs close to negative infinity and zero), and because the bounds focus on the worst case.

Therefore, we chose to focus instead on the average case by attempting to minimize the KL divergence between the original model and the simplified approximation: $\mathrm{KL}(P \| Q) = \sum_x P(x) \log \frac{P(x)}{Q(x)}$ where $P$ is the original network and $Q$ is the simplified approximate network, in which each of $P$'s conditional probability distributions has been simplified. We choose to optimize the KL divergence here because the reverse KL is prone to fitting only a single mode, and we want to avoid excluding any significant parts of the distribution before seeing evidence. Since $Q$'s structure is a subset of $P$'s, we can decompose the KL divergence as follows:

$$\mathrm{KL}(P \| Q) = \sum_i \sum_{\pi_i} P(\pi_i) \sum_{x_i} P(x_i | \pi_i) \log \frac{P(x_i | \pi_i)}{Q(x_i | \pi_i)} \tag{3}$$

where the summation is over all states of the $X_i$'s parents, $\Pi_i$. In other words, the KL divergence can be computed by adding the expected divergence of each local factor, where the expectation is computed according to the global probability distribution. For the case of BNs with tree CPDs (as described in Section 2.1), this means that knowing the distribution of the parent variables allows us to compute the change in KL divergence from pruning a tree CPD.

Unfortunately, computing the distribution of each variable's parents is intractable and must be approximated in some way. We tried two different methods for computing these distributions: estimating the joint parent probabilities from a large number of samples (one million in our experiments) ("P-Samp"), and forming the product of the parent marginals estimated using mean field ("P-MF").

Given a method for computing the parent marginals, we remove the splits that least increase the KL divergence. We implement this by starting from a fully pruned network and greedily adding the splits that most decrease KL divergence. After every 10 splits, we check the number of edges by compiling the candidate network to an AC using the C2D compiler. [1] We stop when the number of edges exceeds our prespecified bound.

### 3.1.2 Learning from samples

The second approach we tried is learning a circuit from a set of generated samples. The samples themselves are generated using forward sampling, in which each variable in the BN is sampled in topological order according to its conditional distribution given its parents. The circuit learning method we chose is the LearnAC algorithm by Lowd and Domingos [13], which greedily learns an AC representing a BN with decision tree CPDs by trading off log likelihood and circuit size. We made one modification to the the LearnAC (LAC) algorithm in order to learn circuits with a fixed number of edges. Instead of using a fixed edge penalty, we start with an edge penalty of 100 and halve it every time we run out of candidate splits with non-negative scores. The effect of this modified procedure is to conservatively selects splits that add few edges to the circuit at first, and become increasingly liberal until the edge limit is reached. Tuning the initial edge penalty can lead to slightly better performance at the cost of additional training time. We also explored using the BN structure to guide the AC structure search (for example, by excluding splits that would violate the partial order of the original BN), but these restrictions offered no significant advantage in accuracy.

Many modifications to this procedure are possible. Larger edge budgets or different heuristics could yield more accurate circuits. With additional engineering, the LearnAC algorithm could be adapted to dynamically request only as many samples as necessary to be confident in its choices. For example, Hulten and Domingos [14] have developed methods that scale learning algorithms to datasets of arbitrary size; the same approach could be used here, except in a "pull" setting where the data is generated on-demand. Spending a long time finding the most accurate circuit may be worthwhile, since the cost is amortized over all queries.

We are not the first to propose sampling as a method for converting intractable models into tractable ones. Wang *et al.* [15] used a similar procedure for learning a latent tree model to approximate a

BN. They found that the learned models had faster or more accurate inference on a wide range of standard BNs (where exact inference is somewhat tractable). In a semi-supervised setting, Liang *et al.* [16] trained a conditional random field (CRF) from a small amount of labeled training data, used the CRF to label additional examples, and learned independent logistic regression models from this expanded dataset.

## 3.2 Parameter optimization

In this section, we describe three methods for selecting AC parameters: forward sampling, variational optimization, and Gibbs sampling.

### 3.2.1 Forward sampling

In $AC^2$-F, we use forward sampling to generate a set of samples from the original BN (one million in our experiments) and maximum likelihood estimation to estimate the AC parameters from those samples. This can be done in closed form because, before conditioning on evidence, the AC structure also represents a BN. $AC^2$-F selects these parameters once per domain, before conditioning on any evidence. This makes it very fast at query time.

$AC^2$-F can be viewed as approximately minimizing the KL divergence $KL(P \parallel Q)$ between the BN distribution $P$ and the AC distribution $Q$. For conditional queries $P(Y|X = x_{ev})$, we are more interested in the divergence of the conditional distributions, $KL(P(.|x_{ev}) \parallel Q(.|x_{ev}))$. The following theorem bounds the conditional KL divergence as a function of the unconditional KL divergence:

**Theorem 1.** *For discrete probability distributions $P$ and $Q$, and evidence $x_{ev}$,*

$$KL(P(.|x_{ev}) \parallel Q(.|x_{ev})) \leq \frac{1}{P(x_{ev})} KL(P \parallel Q)$$

(See the supplementary materials for the proof.) From this theorem, we expect $AC^2$-F to work better when evidence is likely (*i.e.*, $P(x_{ev})$ is not too small). For rare evidence, the conditional KL divergence could be much larger than the unconditional KL divergence.

### 3.2.2 Variational optimization

Since $AC^2$-F selects parameters based on the unconditioned BN, it may do poorly when conditioning on rare evidence. An alternative is to choose AC parameters that (locally) minimize the reverse KL divergence to the BN conditioned on evidence. Let $P$ and $Q$ be log-linear models, i.e.:

$$\log P(x) = -\log Z_P + \sum_i w_i f_i(x) \qquad \log Q(x) = -\log Z_Q + \sum_j v_j g_j(x)$$

The reverse KL divergence and its gradient can now be written as follows:

$$KL(Q \parallel P) = \sum_j v_j E_Q(g_j) - \sum_i w_i E_Q(f_i) + \log \frac{Z_P}{Z_Q} \tag{4}$$

$$\frac{\partial}{\partial v_j} KL(Q \parallel P) = \sum_k v_k (E_Q(g_k g_j) - Q(g_k)Q(g_j)) - \sum_i v_i (E_Q(f_i g_j) - Q(f_i)Q(g_j)) \tag{5}$$

where $E_Q(g_k g_j)$ is the expected value of $g_k(x) \times g_j(x)$ according to $Q$. In our application, $P$ is the BN conditioned on evidence and $Q$ is the AC. Since inference in $Q$ (the AC) is tractable, the gradient can be computed exactly.

We can optimize this using any numerical optimization method, such as gradient descent. Due to local optima, the results may depend on the optimization procedure and its initialization. In experiments, we used the limited memory BFGS algorithm (L-BFGS) [17], initialized with $AC^2$-F.

We now discuss how to compute the gradient efficiently in a circuit with $e$ edges. By setting leaf values and evaluating the circuit as described by Darwiche [1], we can compute the probability of any conjunctive feature $Q(f_i)$ (or $Q(g_k)$) in $O(e)$ operations. If we differentiate the circuit after conditioning on a feature $f_i$ (or $g_k$), we can obtain the probabilities of the conjunctions $Q(f_i g_j)$ (or $Q(g_k g_j)$) for all $g_j$ in $O(e)$ time. Therefore, if there are $n$ features in $P$, and $m$ features in $Q$, then the total complexity of computing the derivative is $O((n + m)e)$. Since there are typically fewer features in $Q$ than $P$, this simplifies to $O(ne)$.

These methods are applicable to any tractable structure represented as an AC, including low treewidth models, mixture models, latent tree models, etc. We refer to this method as $AC^2$-V.

### 3.2.3 Gibbs sampling

While optimizing the reverse KL is a popular choice for approximate inference, there are certain risks. Even if $\text{KL}(Q \| P)$ is small, $Q$ may assign very small or zero probabilities to important modes of $P$. Furthermore, we are only guaranteed to find a local optimum, which may be much worse than the global optimum. The "regular" KL divergence, does not suffer these disadvantages, but is impractical to compute since it involves expectations according to $P$:

$$\text{KL}(P \| Q) = \sum_i w_i E_P(f_i) - \sum_j v_j E_P(g_j) + \log Z_Q/Z_P \qquad (6)$$

$$\frac{\partial}{\partial v_j} \text{KL}(P \| Q) = E_Q(g_j) - E_P(g_j) \qquad (7)$$

Therefore, minimizing $\text{KL}(P \| Q)$ by gradient descent or L-BFGS requires computing the conditional probability of each AC feature according to the BN, $E_P(g_j)$. Note that these only need to be computed once, since they are unaffected by the AC feature weights, $v_j$. We chose to approximate these expectations using Gibbs sampling, but an alternate inference method (e.g., importance sampling) could be substituted. The probabilities of the AC features according to the AC, $E_Q(g_j)$, can be computed in parallel by differentiating the circuit, requiring time $O(e)$.[2] This is typically orders of magnitude faster than the variational approach described above, since each optimization step runs in $O(e)$ instead of $O(ne)$, where $n$ is the number of BN features. We refer to this method as $\text{AC}^2$-G.

## 4 Experiments

In this section, we compare the proposed methods experimentally and demonstrate that approximate compilation is an accurate and efficient technique for inference in intractable networks.

### 4.1 Datasets

We wanted to evaluate our methods on challenging, realistic networks where exact inference is intractable, even for the most sophisticated arithmetic circuit-based techniques. This ruled out most traditional benchmarks, for which ACs can already perform exact inference [9]. We generated intractable networks by learning them from eight real-world datasets using the WinMine Toolkit [18]. The WinMine Toolkit learns BNs with tree-structured CPDs, leading to complex models with high tree-width. In theory, this additional structure can be exploited by existing arithmetic circuit techniques, but in practice, compilation techniques ran out of memory on all eight networks. See Davis and Domingos [19] and our supplementary material for more details on the datasets and the networks learned from them, respectively.

### 4.2 Structure selection

In our first set of experiments, we compared the structure selection algorithms from Section 3.1 according to their ability to approximate the original models. Since computing the KL divergence directly is intractable, we approximated it using random samples $x^{(i)}$:

$$D(P\|Q) = \sum_x P(x) \log \frac{P(x)}{Q(x)} = E_P[\log(P(x)/Q(x))] \approx \frac{1}{m} \sum_i \log(P(x^{(i)})/Q(x^{(i)})) \quad (8)$$

where $m$ is the number of samples (10,000 in our experiments). These samples were distinct from the training data, and the same set of samples was used to evaluate each algorithm.

For LearnAC, we trained circuits with a limit of 100,000 edges. All circuits were learned using 100,000 samples, and then the parameters were set using $\text{AC}^2$-F with 1 million samples.[3] Training time ranged from 17 minutes (KDD Cup) to 8 hours (EachMovie). As an additional baseline, we also learned tree-structured BNs from the same 1 million samples using the Chow-Liu algorithm [20].

Results are in Table 1. The learned arithmetic circuit (LAC) achieves the best performance on all datasets, often by a wide margin. We also observe that, of the pruning methods, samples (P-Samp) work better than mean field marginals (P-MF). Chow-Liu trees (C-L) typically perform somewhere between P-MF and P-Samp. For the rest of this paper, we focus on structures selected by LearnAC.

Table 1: KL divergence of different structure selection algorithms.

|  | P-MF | P-Samp | C-L | LAC |
|---|---|---|---|---|
| KDD Cup | 2.44 | 0.10 | 0.23 | 0.07 |
| Plants | 8.41 | 2.29 | 4.48 | 1.27 |
| Audio | 4.99 | 3.31 | 4.47 | 2.12 |
| Jester | 5.14 | 3.55 | 5.08 | 2.82 |
| Netflix | 3.83 | 3.06 | 4.14 | 2.24 |
| MSWeb | 1.78 | 0.52 | 0.70 | 0.38 |
| Book | 4.90 | 2.43 | 2.84 | 1.89 |
| EachMovie | 29.66 | 17.61 | 17.11 | 11.12 |

Table 2: Mean time for answering a single conditional query, in seconds.

|  | $AC^2$-F | $AC^2$-V | $AC^2$-G | BP | MF | Gibbs |
|---|---|---|---|---|---|---|
| KDD Cup | 0.022 | 3803 | 11.2 | 0.050 | 0.025 | 2.5 |
| Plants | 0.022 | 2741 | 11.2 | 0.081 | 0.073 | 2.8 |
| Audio | 0.023 | 4184 | 14.4 | 0.063 | 0.048 | 3.4 |
| Jester | 0.019 | 3448 | 13.8 | 0.054 | 0.057 | 3.3 |
| Netflix | 0.021 | 3050 | 12.3 | 0.057 | 0.053 | 3.3 |
| MSWeb | 0.022 | 2831 | 12.2 | 0.277 | 0.046 | 4.3 |
| Book | 0.020 | 5190 | 16.1 | 0.864 | 0.059 | 6.6 |
| EachMovie | 0.022 | 10204 | 28.6 | 1.441 | 0.342 | 11.0 |

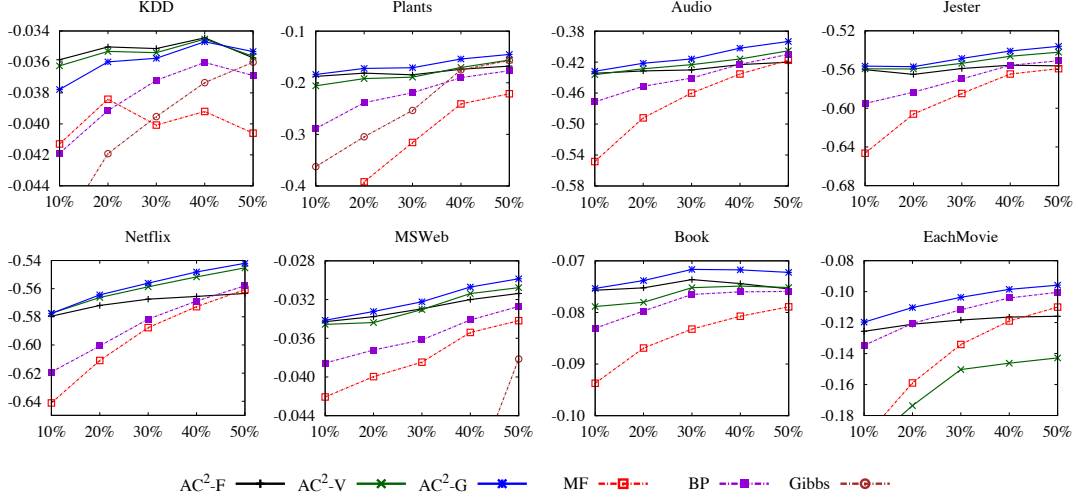

Figure 1: Average conditional log likelihood of the query variables (y axis), divided by the number of query variables (x axis). Higher is better. Gibbs often performs too badly to appear in the frame.

## 4.3 Conditional probabilities

Using structures selected by LearnAC, we compared the accuracy of $AC^2$-F, $AC^2$-V, and $AC^2$-G to mean field (MF), loopy belief propagation (BP), and Gibbs sampling (Gibbs) on conditional probability queries. We ran MF and BP to convergence. For Gibbs sampling, we ran 10 chains, each with 1000 burn-in iterations and 10,000 sampling iterations. All methods exploited CPD structure whenever possible (e.g., in the computation of BP messages). All code will be publicly released.

Since most of these queries are intractable to compute exactly, we cannot determine the true probabilities directly. Instead, we generated 100 random samples from each network, selected a random subset of the variables to use as evidence (10%-50% of the total variables), and measured the log conditional probability of the non-evidence variables according to each inference method. Different queries used different evidence variables. This approximates the KL divergence between the true and inferred conditional distributions up to a constant. We reduced the variance of this approximation by selecting additional queries for each evidence configuration. Specifically, we generated 100,000 samples and kept the ones compatible with the evidence, up to 10,000 per configuration. For some evidence, none of the 100,000 samples were compatible, leaving just the original query.

Full results are in Figure 1. Table 2 contains the average inference time for each method.

Overall, $AC^2$-F does very well against BP and even better against MF and Gibbs, especially with lesser amounts of evidence. Its somewhat worse performance at greater amounts of evidence is consistent with Theorem 1. $AC^2$-F is also the fastest of the inference methods, making it a very good choice for speedy inference with small to moderate amounts of evidence.

$AC^2$-V obtains higher accuracy than $AC^2$-F at higher levels of evidence, but is often less accurate at lesser amounts of evidence. This can be attributed to different optimization and evaluation metrics:

reducing $KL(Q \parallel P)$ may sometimes lead to increased $KL(P \parallel Q)$. On EachMovie, $AC^2$-V does particularly poorly, getting stuck in a worse local optimum than the much simpler MF. $AC^2$-V is also the slowest method, by far.

$AC^2$-G is the most accurate method overall. It dominates BP, MF, and Gibbs on all datasets. With the same number of samples, $AC^2$-G takes 2-4 times longer than Gibbs. This additional running time is partly due to the parameter optimization step and partly due to the fact that $AC^2$-G is computing many expectations in parallel, and therefore has more bookkeeping per sample. If we increase the number of samples in Gibbs by a factor of 10 (not shown), then Gibbs wins on KDD at 40 and 50% and Plants at 50% evidence, but is also significantly slower than $AC^2$-G. Compared to the other AC methods, $AC^2$-G wins everywhere except for KDD at 10-40% evidence and Netflix at 10% evidence. If we increase the number of samples in $AC^2$-G by a factor of 10 (not shown), then it beats $AC^2$-F and $AC^2$-V on every dataset. The running time of $AC^2$-G is split approximately evenly between computing sufficient statistics and optimizing parameters with L-BFGS.

Gibbs sampling did poorly in almost all of the scenarios, which can be attributed to the fact that it is unable to accurately estimate the probabilities of very infrequent events. Most conjunctions of dozens or hundreds of variables are very improbable, even if conditioned on a large amount of evidence. If a certain configuration is never seen, then its probability is estimated to be very low (non-zero due to smoothing). MF and BP did not have this problem, since they represent the conditional distribution as a product of marginals, each of which can be estimated reasonably well. In follow-up experiments, we found that using Gibbs sampling to compute the marginals yielded slightly better accuracy than BP, but much slower. $AC^2$-G can be seen as a generalization of using Gibbs sampling to compute marginals, just as $AC^2$-V generalizes MF.

## 5    Conclusion

Arithmetic circuits are an attractive alternative to junction trees due to their ability to exploit determinism and context-specific independence. However, even with ACs, exact inference remains intractable for many networks of interest. In this paper, we introduced the first approximate compilation methods, allowing us to apply ACs to any BN. Our most efficient method, $AC^2$-F, is faster than traditional approximate inference methods and more accurate most of the time. Our most accurate method, $AC^2$-G, is more accurate than the baselines on every domain.

One of the key lessons is that combining sampling and learning is a good strategy for accurate approximate inference. Sampling generates a coarse approximation of the desired distribution which is subsequently smoothed by learning. For structure selection, an AC learning method applied to samples was more effective than exact compilation of a simplified network. For parameter selection, maximum likelihood estimation applied to Gibbs samples was both faster and more effective than variational inference in ACs.

For future work, we hope to extend our methods to Markov networks, in which generating samples is a difficult inference problem in itself. Similar methods could be used to select AC structures tuned to particular queries, since a BN conditioned on evidence can be represented as a Markov network. This could lead to more accurate results, especially in cases with a lot of evidence, but the cost would no longer be amortized over all future queries. Comparisons with more sophisticated baselines are another important item for future work.

**Acknowledgements**

The authors wish to thank Christopher Meek and Jesse Davis for helpful comments. This research was partly funded by ARO grant W911NF-08-1-0242, AFRL contract FA8750-09-C-0181, DARPA contracts FA8750-05-2-0283, FA8750-07-D-0185, HR0011-06-C-0025, HR0011-07-C-0060 and NBCH-D030010, NSF grants IIS-0534881 and IIS-0803481, and ONR grant N00014-08-1-0670. The views and conclusions contained in this document are those of the authors and should not be interpreted as necessarily representing the official policies, either expressed or implied, of ARO, DARPA, NSF, ONR, or the United States Government.

## Footnotes

[1] Available at http://reasoning.cs.ucla.edu/c2d/.

[2]To support optimization methods that perform line search (including L-BFGS), we can similarly approximate $\text{KL}(P \| Q)$. $\log Z_Q$ can also be computed in $O(e)$ time.

[3]With 1 million samples, we ran into memory limitations that a more careful implementation might avoid.

## References

[1] A. Darwiche. A differential approach to inference in Bayesian networks. *Journal of the ACM*, 50(3):280–305, 2003.

[2] J. Pearl. *Probabilistic Reasoning in Intelligent Systems: Networks of Plausible Inference*. Morgan Kaufmann, San Francisco, CA, 1988.

[3] C. Boutilier, N. Friedman, M. Goldszmidt, and D. Koller. Context-specific independence in Bayesian networks. In *Proc. of the 12th Conference on Uncertainty in Artificial Intelligence*, pages 115–123, Portland, OR, 1996. Morgan Kaufmann.

[4] N. Friedman and M. Goldszmidt. Learning Bayesian networks with local structure. In *Proc. of the 12th Conference on Uncertainty in Artificial Intelligence*, pages 252–262, Portland, OR, 1996. Morgan Kaufmann.

[5] D. Chickering, D. Heckerman, and C. Meek. A Bayesian approach to learning Bayesian networks with local structure. In *Proc. of the 13th Conference on Uncertainty in Artificial Intelligence*, pages 80–89, Providence, RI, 1997. Morgan Kaufmann.

[6] Arthur Choi and Adnan Darwiche. A variational approach for approximating Bayesian networks by edge deletion. In *Proc. of the 22nd Conference on Uncertainty in Artificial Intelligence (UAI-06)*, Arlington, Virginia, 2006. AUAI Press.

[7] E. P. Xing, M. I. Jordan, and S. Russell. Graph partition strategies for generalized mean field inference. In *Proc. of the 20th Conference on Uncertainty in Artificial Intelligence*, pages 602–610, Banff, Canada, 2004.

[8] D. Geiger, C. Meek, and Y. Wexler. A variational inference procedure allowing internal structure for overlapping clusters and deterministic constraints. *Journal of Artificial Intelligence Research*, 27:1–23, 2006.

[9] M. Chavira and A. Darwiche. Compiling Bayesian networks using variable elimination. In *Proc. of the 20th International Joint Conference on Artificial Intelligence (IJCAI)*, pages 2443–2449, 2007.

[10] R. Dechter and R. Mateescu. AND/OR search spaces for graphical models. *Artificial Intelligence*, 171:73–106, 2007.

[11] R. Dechter. Bucket elimination: a unifying framework for reasoning. *Artificial Intelligence*, 113:41–85, 1999.

[12] Y. Wexler and C. Meek. MAS: a multiplicative approximation scheme for probabilistic inference. In *Advances in Neural Information Processing Systems 22*, Cambridge, MA, 2008. MIT Press.

[13] D. Lowd and P. Domingos. Learning arithmetic circuits. In *Proc. of the 24th Conference on Uncertainty in Artificial Intelligence*, Helsinki, Finland, 2008. AUAI Press.

[14] G. Hulten and P. Domingos. Mining complex models from arbitrarily large databases in constant time. In *Proc. of the 8th ACM SIGKDD International Conference on Knowledge Discovery and Data Mining*, pages 525–531, Edmonton, Canada, 2002. ACM Press.

[15] Y. Wang, N. L. Zhang, and T. Chen. Latent tree models and approximate inference in Bayesian networks. *Journal of Artificial Intelligence Research*, 32:879–900, 2008.

[16] P. Liang, III H. Daumé, and D. Klein. Structure compilation: trading structure for features. In *Proc. of the 25th International Conference on Machine Learning*, pages 592–599, Helsinki, Finland, 2008. ACM.

[17] D. C. Liu and J. Nocedal. On the limited memory BFGS method for large scale optimization. *Mathematical Programming*, 45(3):503–528, 1989.

[18] D. M. Chickering. The WinMine toolkit. Technical Report MSR-TR-2002-103, Microsoft, Redmond, WA, 2002.

[19] J. Davis and P. Domingos. Bottom-up learning of Markov network structure. In *Proc. of the 27th International Conference on Machine Learning*, Haifa, Israel, 2010. ACM Press.

[20] C. K. Chow and C. N Liu. Approximating discrete probability distributions with dependence trees. *IEEE Transactions on Information Theory*, 14:462–467, 1968.

